# Accelerated Variational Dirichlet Process Mixtures

**Kenichi Kurihara**
Dept. of Computer Science
Tokyo Institute of Technology
Tokyo, Japan
`kurihara@mi.cs.titech.ac.jp`

**Max Welling**
Bren School of Information and Computer Science
UC Irvine
Irvine, CA 92697-3425
`welling@ics.uci.edu`

**Nikos Vlassis**
Informatics Institute
University of Amsterdam
The Netherlands
`vlassis@science.uva.nl`

## Abstract

Dirichlet Process (DP) mixture models are promising candidates for clustering applications where the number of clusters is unknown a priori. Due to computational considerations these models are unfortunately unsuitable for large scale data-mining applications. We propose a class of deterministic accelerated DP mixture models that can routinely handle millions of data-cases. The speedup is achieved by incorporating kd-trees into a variational Bayesian algorithm for DP mixtures in the stick-breaking representation, similar to that of Blei and Jordan (2005). Our algorithm differs in the use of kd-trees and in the way we handle truncation: we only assume that the variational distributions are fixed at their priors after a certain level. Experiments show that speedups relative to the standard variational algorithm can be significant.

## 1 Introduction

Evidenced by three recent workshops[1], nonparametric Bayesian methods are gaining popularity in the machine learning community. In each of these workshops computational efficiency was mentioned as an important direction for future research. In this paper we propose computational speedups for Dirichlet Process (DP) mixture models [1, 2, 3, 4, 5, 6, 7], with the purpose of improving their applicability in modern day data-mining problems where millions of data-cases are no exception.

Our approach is related to, and complements, the variational mean-field algorithm for DP mixture models of Blei and Jordan [7]. In this approach, the intractable posterior of the DP mixture is approximated with a factorized variational finite (truncated) mixture model with $T$ components, that is optimized to minimize the KL distance to the posterior. However, a downside of their model is that the variational families are not nested over $T$, and locating an optimal truncation level $T$ may be difficult (see Section 3).

In this paper we propose an alternative variational mean-field algorithm, called VDP (Variational DP), in which the variational families are nested over $T$. In our model we allow for an unbounded number of components for the variational mixture, but we *tie* the variational distributions after level

http://www.cs.toronto.edu/~beal/npbayes/
http://www2.informatik.hu-berlin.de/~bickel/npb-workshop.html

$T$ to their priors. Our algorithm proceeds in a greedy manner by starting with $T = 1$ and releasing components when this improves (significantly) the KL bound. Releasing is most effectively done by splitting a component in two children and updating them to convergence. Our approach essentially resolves the issue in [7] of searching for an optimal truncation level of the variational mixture (see Section 4).

Additionally, a significant contribution is that we incorporate kd-trees into the VDP algorithm as a way to speed up convergence [8, 9]. A kd-tree structure recursively partitions the data space into a number of nodes, where each node contains a subset of the data-cases. Following [9], for a given tree expansion we tie together the responsibility over mixture components of all data-cases contained in each outer node of the tree. By caching certain sufficient statistics in each node of the kd-tree we then achieve computational gains, while the variational approximation becomes a function of the depth of the tree at which one operates (see Section 6).

The resulting Fast-VDP algorithm provides an elegant way to trade off computational resources against accuracy. We can always release new components from the pool and split kd-tree nodes as long as we have computational resources left. Our setup guarantees that this will always (at least in theory) improve the KL bound (in practice local optima may force us to reject certain splits, see Section 7). As we empirically demonstrate in Section 8, a kd-tree can offer significant speedups, allowing our algorithm to handle millions of data-cases. As a result, Fast-VDP is the first algorithm entertaining an unbounded number of clusters that is practical for modern day data-mining applications.

## 2  The Dirichlet Process Mixture in the Stick-Breaking Representation

A DP mixture model in the stick-breaking representation can be viewed as possessing an infinite number of components with random mixing weights [4]. In particular, the generative model of a DP mixture assumes:

- An infinite collection of components $H = \{\eta_i\}_{i=1}^{\infty}$ that are independently drawn from a prior $p_\eta(\eta_i|\lambda)$ with hyperparameters $\lambda$.
- An infinite collection of 'stick lengths' $V = \{v_i\}_{i=1}^{\infty}$, $v_i \in [0,1]$, $\forall i$, that are independently drawn from a prior $p_v(v_i|\alpha)$ with hyperparameters $\alpha$. They define the mixing weights $\{\pi_i\}_{i=1}^{\infty}$ of the mixture as $\pi_i(V) = v_i \prod_{j=1}^{i-1}(1-v_j)$, for $i = 1, \ldots, \infty$.
- An observation model $p_x(x|\eta)$ that generates a datum $x$ from component $\eta$.

Given a dataset $X = \{x_n\}_{n=1}^{N}$, each data-case $x_n$ is assumed to be generated by first drawing a component label $z_n = k \in \{1, \ldots, \infty\}$ from the infinite mixture with probability $p_z(z_n = k|V) \equiv \pi_k(V)$, and then drawing $x_n$ from the corresponding observation model $p_x(x_n|\eta_k)$.

We will denote $Z = \{z_n\}_{n=1}^{N}$ the set of all labels, $W = \{H, V, Z\}$ the set of all latent variables of the DP mixture, and $\theta = \{\lambda, \alpha\}$ the hyperparameters. In clustering problems we are mainly interested in computing the posterior over data labels $p(z_n|X, \theta)$, as well as the predictive density $p(x|X, \theta) = \int_{H,V} p(x|H, V) \int_Z p(W|X, \theta)$, which are both intractable since $p(W|X, \theta)$ cannot be computed analytically.

## 3  Variational Inference in Dirichlet Process Mixtures

For variational inference, the intractable posterior $p(W|X, \theta)$ of the DP mixture can be approximated with a parametric family of factorized variational distributions $q(W; \phi)$ of the form

$$q(W; \phi) = \prod_{i=1}^{L} \left[ q_{v_i}(v_i; \phi_i^v) \, q_{\eta_i}(\eta_i; \phi_i^\eta) \right] \prod_{n=1}^{N} q_{z_n}(z_n) \tag{1}$$

where $q_{v_i}(v_i; \phi_i^v)$ and $q_{\eta_i}(\eta_i; \phi_i^\eta)$ are parametric models with parameters $\phi_i^v$ and $\phi_i^\eta$ (one parameter per $i$), and $q_{z_n}(z_n)$ are discrete distributions over the component labels (one distribution per $n$). Blei and Jordan [7] define an explicit truncation level $L \equiv T$ for the variational mixture in (1) by setting $q_{v_T}(v_T = 1) = 1$ and assuming that data-cases assign zero responsibility to components

with index higher than the truncation level $T$, i.e., $q_{z_n}(z_n > T) = 0$. Consequently, in their model only components of the mixture up to level $T$ need to be considered. Variational inference then consists in estimating a set of $T$ parameters $\{\phi_i^v, \phi_i^\eta\}_{i=1}^T$ and a set of $N$ distributions $\{q_{z_n}(z_n)\}_{n=1}^N$, collectively denoted by $\phi$, that minimize the Kullback-Leibler divergence $\mathrm{D}[q(W; \phi)||p(W|X, \theta)]$ between the true posterior and the variational approximation, or equivalently that minimize the free energy $F(\phi) = E_q[\log q(W; \phi)] - E_q[\log p(W, X|\theta)]$. Since each distribution $q_{z_n}(z_n)$ has nonzero support only for $z_n \leq T$, minimizing $F(\phi)$ results in a set of update equations for $\phi$ that involve only finite sums [7].

However, explicitly truncating the variational mixture as above has the undesirable property that the variational family with truncation level $T$ is not contained within the variational family with truncation level $T + 1$, i.e., the families are not nested.[2] The result is that there may be an optimal *finite* truncation level $T$ for $q$, that contradicts the intuition that the more components we allow in $q$ the better the approximation should be (reaching its best when $T \to \infty$). Moreover, locating a near-optimal truncation level may be difficult since $F$ as a function of $T$ may exhibit local minima (see Fig. 4 in [7]).

## 4  Variational Inference with an Infinite Variational Model

Here we propose a slightly different variational model for $q$ that allows families over $T$ to be nested. In our setup, $q$ is given by (1) where we let $L$ go to infinity but we *tie* the parameters of all models after a specific level $T$ (with $T \ll L$). In particular, we impose the condition that for all components with index $i > T$ the variational distributions for the stick-lengths $q_{v_i}(v_i)$ and the variational distributions for the components $q_{\eta_i}(\eta_i)$ are equal to their corresponding priors, i.e., $q_{v_i}(v_i; \phi_i^v) = p_v(v_i|\alpha)$ and $q_{\eta_i}(\eta_i; \phi_i^\eta) = p_\eta(\eta_i|\lambda)$. In our model we define the free energy $F$ as the limit $F = \lim_{L \to \infty} F_L$, where $F_L$ is the free energy defined by $q$ in (1) and a truncated DP mixture at level $L$ (justified by the almost sure convergence of an $L$-truncated Dirichlet process to an infinite Dirichlet process when $L \to \infty$ [6]). Using the parameter tying assumption for $i > T$, the free energy reads

$$F = \sum_{i=1}^T \left\{ E_{q_{v_i}}\left[\log \frac{q_{v_i}(v_i; \phi_i^v)}{p_v(v_i|\alpha)}\right] + E_{q_{\eta_i}}\left[\log \frac{q_{\eta_i}(\eta_i; \phi_i^\eta)}{p_\eta(\eta_i|\lambda)}\right] \right\} + \sum_{n=1}^N E_q\left[\log \frac{q_{z_n}(z_n)}{p_z(z_n|V)p_x(x_n|\eta_{z_n})}\right]. \quad (2)$$

In our scheme $T$ defines an implicit truncation level of the variational mixture, since there are no free parameters to optimize beyond level $T$. As in [7], the free energy $F$ is a function of $T$ parameters $\{\phi_i^v, \phi_i^\eta\}_{i=1}^T$ and $N$ distributions $\{q_{z_n}(z_n)\}_{n=1}^N$. However, contrary to [7], data-cases may now assign *nonzero* responsibility to components beyond level $T$, and therefore each $q_{z_n}(z_n)$ must now have infinite support (which requires computing infinite sums in the various quantities of interest). An important implication of our setup is that the variational families are now nested with respect to $T$ (since for $i > T$, $q_{v_i}(v_i)$ and $q_{\eta_i}(\eta_i)$ can always revert to their priors), and as a result it is guaranteed that as we increase $T$ there exist solutions that decrease $F$. This is an important result because it allows for optimization with adaptive $T$ starting from $T = 1$ (see Section 7).

¿From the last term of (2) we directly see that the $q_{z_n}(z_n)$ that minimizes $F$ is given by

$$q_{z_n}(z_n = i) = \frac{\exp(S_{n,i})}{\sum_{j=1}^\infty \exp(S_{n,j})} \quad (3)$$

where

$$S_{n,i} = E_{q_V}[\log p_z(z_n = i|V)] + E_{q_{\eta_i}}[\log p_x(x_n|\eta_i)]. \quad (4)$$

Minimization of $F$ over $\phi_i^v$ and $\phi_i^\eta$ can be carried out by direct differentiation of (2) for particular choices of models for $q_{v_i}$ and $q_{\eta_i}$ (see Section 5). Using $q_{z_n}$ from (3), the free energy (2) reads

$$F = \sum_{i=1}^T \left\{ E_{q_{v_i}}\left[\log \frac{q_{v_i}(v_i; \phi_i^v)}{p_v(v_i|\alpha)}\right] + E_{q_{\eta_i}}\left[\log \frac{q_{\eta_i}(\eta_i; \phi_i^\eta)}{p_\eta(\eta_i|\lambda)}\right] \right\} - \sum_{n=1}^N \log \sum_{i=1}^\infty \exp(S_{n,i}). \quad (5)$$

Evaluation of $F$ requires computing the infinite sum $\sum_{i=1}^\infty \exp(S_{n,i})$ in (5). The difficult part is $\sum_{i=T+1}^\infty \exp(S_{n,i})$. Under the parameter tying assumption for $i > T$, most terms of $S_{n,i}$ in (4)

factor out of the infinite sum as constants (since they do not depend on $i$) except for the term $\sum_{j=T+1}^{i-1} E_{p_v}[\log(1-v)] = (i-1-T)E_{p_v}[\log(1-v)]$. From the above, the infinite sum can be shown to be

$$\sum_{i=T+1}^{\infty} \exp(S_{n,i}) = \frac{S_{n,T+1}}{1 - \exp\left(E_{p_v}[\log(1-v)]\right)}. \tag{6}$$

Using the variational $q(W)$ as an approximation to the true posterior $p(W|X,\theta)$, the required posterior over data labels can be approximated by $p(z_n|X,\theta) \approx q_{z_n}(z_n)$. Although $q_{z_n}(z_n)$ has infinite support, in practice it suffices to use the individual $q_{z_n}(z_n = i)$ for the finite part $i \leq T$, and the cumulative $q_{z_n}(z_n > T)$ for the infinite part. Finally, using the parameter tying assumption for $i > T$, and the identity $\sum_{i=1}^{\infty} \pi_i(V) = 1$, the predictive density $p(x|X,\theta)$ can be approximated by

$$p(x|X,\theta) \approx \sum_{i=1}^{T} E_{q_V}[\pi_i(V)]E_{q_{\eta_i}}[p_x(x|\eta_i)] + \left[1 - \sum_{i=1}^{T} E_{p_v}[\pi_i(V)]\right] E_{p_\eta}[p_x(x|\eta)]. \tag{7}$$

Note that all quantities of interest, such as the free energy (5) and the predictive distribution (7), can be computed analytically even though they involve infinite sums.

## 5 Solutions for the exponential family

The results in the previous section apply independently of the choice of models for the DP mixture. In this section we provide analytical solutions for models in the exponential family. In particular we assume that $p_v(v_i|\alpha) = \text{Beta}(\alpha_1, \alpha_2)$ and $q_{v_i}(v_i; \phi_i^v) = \text{Beta}(\phi_{i,1}^v, \phi_{i,2}^v)$, and that $p_x(x|\eta)$, $p_\eta(\eta|\lambda)$, and $q_{\eta_i}(\eta_i; \phi_i^\eta)$ are given by

$$p_x(x|\eta) = h(x)\exp\{\eta^T x - a(\eta)\} \tag{8}$$

$$p_\eta(\eta|\lambda) = h(\eta)\exp\{\lambda_1\eta + \lambda_2(-a(\eta)) - a(\lambda)\} \tag{9}$$

$$q_{\eta_i}(\eta_i; \phi_i^\eta) = h(\eta_i)\exp\{\phi_{i,1}^\eta\eta_i + \phi_{i,2}^\eta(-a(\eta_i)) - a(\phi_i^\eta)\}. \tag{10}$$

In this case, the probabilities $q_{z_n}(z_n = i)$ are given by (3) with $S_{n,i}$ computed from (4) using

$$E_{q_{v_i}}[\log v_i] = \Psi(\phi_{i,1}^v) - \Psi(\phi_{i,1}^v + \phi_{i,2}^v) \tag{11}$$

$$E_{q_{v_j}}[\log(1-v_j)] = \Psi(\phi_{i,2}^v) - \Psi(\phi_{i,1}^v + \phi_{i,2}^v) \tag{12}$$

$$E_{q_{\eta_i}}[\log p_x(x_n|\eta_i)] = E_{q_{\eta_i}}[\eta_i]^T x_n - E_{q_{\eta_i}}[a(\eta_i)] \tag{13}$$

where $\Psi(\cdot)$ is the digamma function. The optimal parameters $\phi^v, \phi^\eta$ can be found to be

$$\phi_{i,1}^v = \alpha_1 + \sum_{n=1}^{N} q_{z_n}(z_n = i) \qquad \phi_{i,2}^v = \alpha_2 + \sum_{n=1}^{N}\sum_{j=i+1}^{\infty} q_{z_n}(z_n = j) \tag{14}$$

$$\phi_{i,1}^\eta = \lambda_1 + \sum_{n=1}^{N} q_{z_n}(z_n = i)x_n \qquad \phi_{i,2}^\eta = \lambda_2 + \sum_{n=1}^{N} q_{z_n}(z_n = i). \tag{15}$$

The update equations are similar to those in [7] except that we have used $\text{Beta}(\alpha_1, \alpha_2)$ instead of $\text{Beta}(1, \alpha)$, and $\phi_{i,2}^v$ involves an infinite sum $\sum_{j=i+1}^{\infty} q_{z_n}(z_n = j)$ which can be computed using (3) and (6). In [7] the corresponding sum is finite since $q_{z_n}(z_n)$ is truncated at $T$.

Note that the VDP algorithm operates in a space where component labels are distinguishable, i.e., if we permute the labels the total probability of the data changes. Since the *average* a priori mixture weights of the components are ordered by their size, the optimal labelling of the a posteriori variational components is also ordered according to cluster size. Hence, we have incorporated a re-ordering step of components according to approximate size after each optimization step in our final algorithm (a feature that was not present in [7]).

## 6 Accelerating inference using a kd-tree

In this section we show that we can achieve accelerated inference for large datasets when we store the data in a kd-tree [10] and cache data sufficient statistics in each node of the kd-tree [8]. A kd-tree

is a binary tree in which the root node contains all points, and each child node contains a subset of the data points contained in its father node, where points are separated by a (typically axis-aligned) hyperplane. Each point in the set is contained in exactly one node, and the set of outer nodes of a given expansion of the kd-tree form a partition of the data set.

Suppose the kd-tree containing our data $X$ is expanded to some level. Following [9], to achieve accelerated update equations we constrain all $x_n$ in outer node $A$ to share the same $q_{z_n}(z_n) \equiv q_{z_A}(z_A)$. We can then show that, under this constraint, the $q_{z_A}(z_A)$ that minimizes $F$ is given by

$$q_{z_A}(z_A = i) = \frac{\exp(S_{A,i})}{\sum_{j=1}^{\infty} \exp(S_{A,j})} \tag{16}$$

where $S_{A,i}$ is computed as in (4) using (11)–(13) with (13) replaced by $E_{q_{\eta_i}}[\eta_i]^T \langle x \rangle_A - E_{q_{\eta_i}}[a(\eta_i)]$, and $\langle x \rangle_A$ denotes average over all data $x_n$ contained in node $A$. Similarly, if $|n_A|$ is the number of data in node $A$, the optimal parameters can be shown to be

$$\phi_{i,1}^v = \alpha_1 + \sum_A |n_A| q_{z_A}(z_A = i) \qquad\qquad \phi_{i,2}^v = \alpha_2 + \sum_A |n_A| \sum_{j=i+1}^{\infty} q_{z_A}(z_A = j) \tag{17}$$

$$\phi_{i,1}^\eta = \lambda_1 + \sum_A |n_A| q_{z_A}(z_A = i) \langle x \rangle_A \qquad\qquad \phi_{i,2}^\eta = \lambda_2 + \sum_A |n_A| q_{z_A}(z_A = i). \tag{18}$$

Finally, using $q_{z_A}(z_A)$ from (16) the free energy (5) reads

$$F = \sum_{i=1}^{T} \left\{ E_{q_{v_i}} \left[ \log \frac{q_{v_i}(v_i; \phi_i^v)}{p_v(v_i|\alpha)} \right] + E_{q_{\eta_i}} \left[ \log \frac{q_{\eta_i}(\eta_i; \phi_i^\eta)}{p_\eta(\eta_i|\lambda)} \right] \right\} - \sum_A |n_A| \log \sum_{i=1}^{\infty} \exp(S_{A,i}). \tag{19}$$

The infinite sums in (17) and (19) can be computed from (6) with $S_{n,T+1}$ replaced by $S_{A,T+1}$. Note that the cost of each update cycle is $O(T|A|)$, which can be a significant improvement over the $O(TN)$ cost when not using a kd-tree. (The cost of building the kd-tree is $O(N \log N)$ but this is amortized by multiple optimization steps.) Note that by refining the tree (expanding outer nodes) the free energy $F$ cannot increase. This allows us to control the trade-off between computational resources and approximation: we can always choose to descend the tree until our computational resources run out, and the level of approximation will be directly tied to $F$ (deeper levels will mean lower $F$).

# 7 The algorithm

The proposed framework is quite general and allows flexibility in the design of an algorithm. Below we show in pseudocode the algorithm that we used in our experiments (for DP Gaussian mixtures). Input is a dataset $X = \{x_n\}_{n=1}^{N}$ that is already stored in a kd-tree structure. Output is a set of parameters $\{\phi_i^v, \phi_i^\eta\}_{i=1}^{T}$ and a value for $T$. From that we can compute responsibilities $q_{z_n}$ using (3).

---

1. Set $T = 1$. Expand the kd-tree to some initial level (e.g. four).
2. Sample a number of 'candidate' components $c$ according to size $\sum_A |n_A| q_{z_A}(z_A = c)$, and split the component that leads to the maximal reduction of $F_T$. For each candidate $c$ do:
   (a) Expand one-level deeper the outer nodes of the kd-tree that assign to $c$ the highest responsibility $q_{z_A}(z_A = c)$ among all components.
   (b) Split $c$ in two components, $i$ and $j$, through the bisector of its principal component. Initialize the responsibilities $q_{z_A}(z_A = i)$ and $q_{z_A}(z_A = j)$.
   (c) Update only $S_{A,i}, \phi_i^v, \phi_i^\eta$ and $S_{A,j}, \phi_j^v, \phi_j^\eta$ for the new components $i$ and $j$, keeping all other parameters as well as the kd-tree expansion fixed.
3. Update $S_{A,t}, \phi_t^v, \phi_t^\eta$ for all $t \leq T + 1$, while expanding the kd-tree and reordering components.
4. If $F_{T+1} > F_T - \epsilon$ then halt, else set $T := T + 1$ and go to step 2.

---

In the above algorithm, the number of sampled candidate components in step 2 can be tuned according to the desired cost/accuracy tradeoff. In our experiments we used 10 candidate components. In step 2b we initialized the responsibilities by $q_{z_A}(z_A = i) = 1 = 1 - q_{z_A}(z_A = j)$ if $\langle x \rangle_A$ is

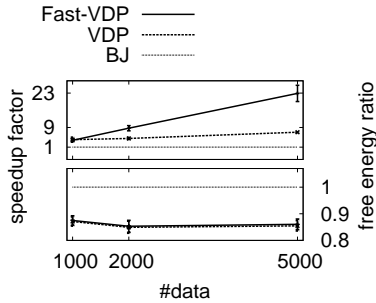

Figure 1: Relative runtimes and free energies of Fast-VDP, VDP, and BJ.

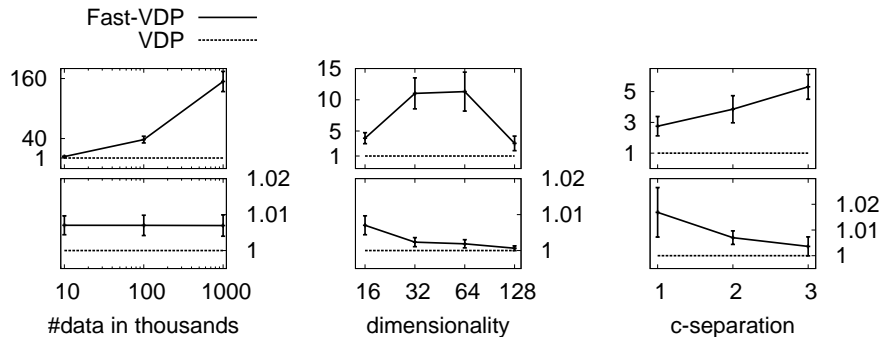

Figure 2: Speedup factors and free energy ratios between Fast-VDP and VDP. Top and bottom figures show speedups and free energy ratios, respectively.

closer to $i$ than to $j$ (according to distance to the expected first moment). In order to speed up the partial updates in step 2c, we additionally set $q_{z_A}(z_A = k) = 0$ for all $k \neq i, j$ (so all responsibility is shared between the two new components). In step 3 we reordered components every one cycle and expanded the kd-tree every three update cycles, controlling the expansion by the relative change of $q_{z_A}(z_A)$ between a node and its children (alternatively one can measure the change of $F_{T+1}$). Finally, in step 2c we monitored convergence of the partial updates through $F_{T+1}$ which can be efficiently computed by adding/subtracting terms involving the new/old components.

## 8 Experimental results

In this section we demonstrate VDP, and its kd-tree extension Fast-VDP, on synthetic and real datasets. In all experiments we assumed a Gaussian observation model $p_x(x|\eta)$ and a Gaussian-inverse Wishart for $p_\eta(\eta|\lambda)$ and $q_{\eta_i}(\eta_i; \phi_i^\eta)$.

**Synthetic datasets.** As argued in Section 4, an important advantage of VDP over the 'BJ' algorithm of [7] is that in VDP the variational families are nested over $T$, which ensures that the free energy is a monotone decreasing function of $T$ and therefore allows for an adaptive $T$ (starting with the trivial initialization $T = 1$). On the contrary, BJ optimizes the parameters for fixed $T$ (and potentially minimizes the resulting free energy over different values of $T$), which requires a nontrivial initialization step for each $T$. Clearly, both the total runtime as well as the quality of the final solution of BJ depend largely on its initialization step, which makes the direct comparison of VDP with BJ difficult. Still, to get a feeling of the relative performance of VDP, Fast-VDP, and BJ, we applied all three algorithms on a synthetic dataset containing 1000 to 5000 data-cases sampled from 10 Gaussians in 16 dimensions, in which the free parameters of BJ were set exactly as described in [7] (20 initialization trials and $T = 20$). VDP and Fast-VDP were also executed until $T = 20$. In Fig. 1 we show the speedup factors and free energy ratios[3] among the three algorithms. Fast-VDP

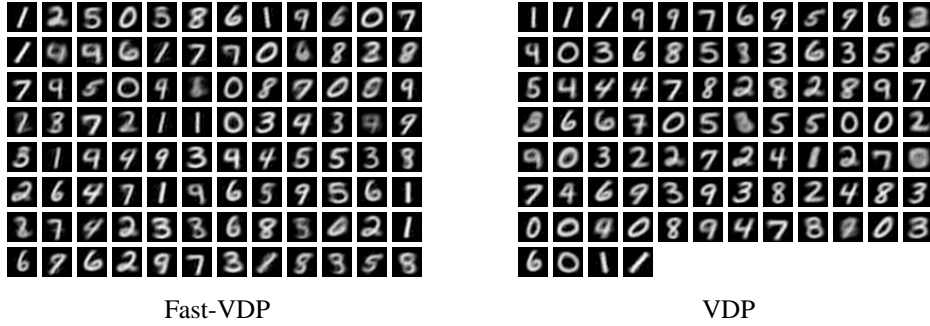

Fast-VDP                                    VDP

Figure 3: Clustering results of Fast-VDP and VDP, with a speedup of 21. The clusters are ordered according to size (from top left to bottom right).

was approximately 23 times faster than BJ, and three times faster than VDP on 5000 data-cases. Moreover, Fast-VDP and VDP were always better than BJ in terms of free energy.

In a second synthetic set of experiments we compared the speedup of Fast-VDP over VDP. We sampled data from 10 Gaussians in dimension $D$ with component separation[4] $c$. Using default number of data-cases $N = 10,000$, dimensionality $D = 16$, and separation $c = 2$, we varied each of them, one at a time. In Fig. 2 we show the speedup factor (top) and the free energy ratio (bottom) between the two algorithms. Note that the latter is always worse for Fast-VDP since it is an approximation to VDP (ratio closer to one means better approximation). Fig. 2-left illustrates that the speedup of Fast-VDP over VDP is at least linear in $N$, as expected from the update equations in Section 6. The speedup factor was approximately 154 for one million data-cases, while the free energy ratio was almost constant over $N$. Fig. 2-center shows an interesting dependence of speed on dimensionality, with $D = 64$ giving the largest speedup. The three plots in Fig. 2 are in agreement with similar plots in [8, 9].

**Real datasets.** In this experiment we applied VDP and Fast-VDP for clustering image data. We used the MNIST dataset (http://yann.lecun.com/exdb/mnist/) which consists of $60,000$ images of the digits 0–9 in 784 dimensions (28 by 28 pixels). We first applied PCA to reduce the dimensionality of the data to 50. Fast-VDP found 96 clusters in $3,379$ seconds with free energy $F = 1.759 \times 10^7$, while VDP found 88 clusters in $72,037$ seconds with free energy $1.684 \times 10^7$. The speedup was 21 and the free energy ratio was $1.044$. The mean images of the discovered components are illustrated in Fig. 3. The results of the two algorithms seem qualitatively similar, while Fast-VDP computed its results much faster than VDP.

In a second real data experiment we clustered documents from citeseer (http://citeseer.ist.psu.edu). The dataset has $30,696$ documents, with a vocabulary size of $32,473$ words. Each document is represented by the counts of words in its abstract. We preprocessed the dataset by Latent Dirichlet Allocation [12] with 200 topics[5]. We subsequently transformed these topic-counts $y_{j,k}$ (count value of $k$'th topic in the $j$'th document) into $x_{j,k} = \log(1 + y_{j,k})$ to fit a normal distribution better. In this problem the elapsed time of Fast-VDP and VDP were 335 seconds and 2,256 seconds, respectively, hence a speedup of $6.7$. The free energy ratio was $1.040$. Fast-VDP found five clusters, while VDP found six clusters. Table 1 shows the three most frequent topics in each cluster. Although the two algorithms found a different number of clusters, we can see that the clusters B and F found by VDP are similar clusters, whereas Fast-VDP did not distinguish between these two. Table 2 shows words included in these topics, showing that the documents are well-clustered.

## 9   Conclusions

We described VDP, a variational mean-field algorithm for Dirichlet Process mixtures, and its fast extension Fast-VDP that utilizes kd-trees to achieve speedups. Our contribution is twofold: First,

| cluster (in descending order) | | Fast-VDP | | | | | VDP | | | | | |
|---|---|---|---|---|---|---|---|---|---|---|---|---|
| | | a | b | c | d | e | A | B | C | D | E | F |
| the three most | 1 | 81 | 73 | 35 | 49 | 76 | 81 | 73 | 35 | 76 | 49 | 73 |
| frequent topics | 2 | 102 | 174 | 50 | 92 | 4 | 102 | 40 | 50 | 4 | 92 | 174 |
| | 3 | 59 | 40 | 110 | 94 | 129 | 59 | 174 | 110 | 129 | 94 | 40 |

Table 1: The three most frequent topics in each clusters. Fast-VDP found five clusters, a–e, while VDP found six clusters, A–F.

| cluster | the most frequent topic | words |
|---|---|---|
| a, A | 81 | economic, policy, countries, bank, growth, firm, public, trade, market, ... |
| b, B, F | 73 | traffic, packet, tcp, network, delay, rate, bandwidth, buffer, end, loss, ... |
| c, C | 35 | algebra, algebras, ring, algebraic, ideal, field, lie, group, theory, ... |
| d, E | 49 | motion, tracking, camera, image, images, scene, stereo, object, ... |
| e, D | 76 | grammar, semantic, parsing, syntactic, discourse, parser, linguistic, ... |

Table 2: Words in the most frequent topic of each cluster.

we extended the framework of [7] to allow for nested variational families and an adaptive truncation level for the variational mixture. Second, we showed how kd-trees can be employed in the framework offering significant speedups, thus extending related results for finite mixture models [8, 9]. To our knowledge, the VDP algorithm is the first nonparametric Bayesian approach to large-scale data mining. Future work includes extending our approach to other models in the stick-breaking representation (e.g., priors of the form $p_{v_i}(v_i|a_i, b_i) = \text{Beta}(a_i, b_i)$), as well as alternative DP mixture representations such as the Chinese restaurant process [3].

**Acknowledgments**

We thank Dave Newman for sharing code and David Blei for helpful comments. This material is based upon work supported by ONR under Grant No. N00014-06-1-0734 and the National Science Foundation under Grant No. 0535278

## Footnotes

[1] http://aluminum.cse.buffalo.edu:8079/npbayes/nipsws05/topics

[2]We thank David Blei for pointing this out.

[3]Free energy ratio is defined as $1 + (F_A - F_B)/|F_B|$, where $A$ and $B$ are either Fast-VDP, VDP or BJ.

[4]A Gaussian mixture is $c$-separated if for each pair $(i,j)$ of components we have $||m_i - m_j||^2 \geq c^2 D \max(\lambda_i^{\max}, \lambda_j^{\max})$, where $\lambda^{\max}$ denotes the maximum eigenvalue of their covariance [11].

[5]We thank David Newman for this preprocessing.

# References

[1] T. Ferguson. A Bayesian analysis of some nonparametric problems. *Ann. Statist.*, 1:209–230, 1973.

[2] C. Antoniak. Mixtures of Dirichlet processes with applications to Bayesian nonparametric problems. *Ann. Statist.*, 2(6):1152–1174, 1974.

[3] D. Aldous. Exchangeability and related topics. In *École d' été de Probabilité de Saint-Flour XIII*, 1983.

[4] J. Sethuraman. A constructive definition of Dirichlet priors. *Statist. Sinica*, 4:639–650, 1994.

[5] C.E. Rasmussen. The infinite Gaussian mixture model. In *NIPS 12*. MIT Press, 2000.

[6] H. Ishwaran and M. Zarepour. Exact and approximate sum-representations for the Dirichlet process. *Can. J. Statist.*, 30:269–283, 2002.

[7] D.M. Blei and M.I. Jordan. Variational inference for Dirichlet process mixtures. *Journal of Bayesian Analysis*, 1(1):121–144, 2005.

[8] A.W. Moore. Very fast EM-based mixture model clustering using multiresolution kd-trees. In *NIPS 11*. MIT Press, 1999.

[9] J.J. Verbeek, J.R.J. Nunnink, and N. Vlassis. Accelerated EM-based clustering of large data sets. *Data Mining and Knowledge Discovery*, 13(3):291–307, 2006.

[10] J.L. Bentley. Multidimensional binary search trees used for associative searching. *Commun. ACM*, 18(9):509–517, 1975.

[11] S. Dasgupta. Learning mixtures of Gaussians. In *IEEE Symp. on Foundations of Computer Science*, 1999.

[12] D.M. Blei, A.Y. Ng, and M.I. Jordan. Latent Dirichlet allocation. *Journal of Machine Learning Research*, 3:993–1022, 2003.
